# Learning Complex Boolean Functions: Algorithms and Applications

**Arlindo L. Oliveira** and **Alberto Sangiovanni-Vincentelli**
Dept. of EECS
UC Berkeley
Berkeley CA 94720

## Abstract

The most commonly used neural network models are not well suited to direct digital implementations because each node needs to perform a large number of operations between floating point values. Fortunately, the ability to learn from examples and to generalize is not restricted to networks of this type. Indeed, networks where each node implements a simple Boolean function (Boolean networks) can be designed in such a way as to exhibit similar properties. Two algorithms that generate Boolean networks from examples are presented. The results show that these algorithms generalize very well in a class of problems that accept compact Boolean network descriptions. The techniques described are general and can be applied to tasks that are not known to have that characteristic. Two examples of applications are presented: image reconstruction and hand-written character recognition.

## 1 Introduction

The main objective of this research is the design of algorithms for empirical learning that generate networks suitable for digital implementations. Although threshold gate networks can be implemented using standard digital technologies, for many applications this approach is expensive and inefficient. Pulse stream modulation [Murray and Smith, 1988] is one possible approach, but is limited to a relatively small number of neurons and becomes slow if high precision is required. Dedicated

boards based on DSP processors can achieve very high performance and are very flexible but may be too expensive for some applications.

The algorithms described in this paper accept as input a training set and generate networks where each node implements a relatively simple Boolean function. Such networks will be called Boolean networks. Many applications can benefit from such an approach because the speed and compactness of digital implementations is still unmatched by its analog counterparts. Additionally, many alternatives are available to designers that want to implement Boolean networks, from full-custom design to field programmable gate arrays. This makes the digital alternative more cost effective than solutions based on analog designs.

Occam's razor [Blumer *et al.*, 1987; Rissanen, 1986] provides the theoretical foundation for the development of algorithms that can be used to obtain Boolean networks that generalize well. According to this paradigm, simpler explanations for the available data have higher predictive power. The induction problem can therefore be posed as an optimization problem: **given a labeled training set, derive the less complex Boolean network that is consistent[1] with the training set.**

Occam's razor, however, doesn't help in the choice of the particular way of measuring complexity that should be used. In general, different types of problems may require different complexity measures. The algorithms described in section 3.1 and 3.2 are greedy algorithms that aim at minimizing one specific complexity measure: the size of the overall network. Although this particular way of measuring complexity may prove inappropriate in some cases, we believe the approach proposed can be generalized and used with minor modifications in many other tasks. The problem of finding the smallest Boolean network consistent with the training set is NP-hard [Garey and Johnson, 1979] and cannot be solved exactly in most cases. Heuristic approaches like the ones described are therefore required.

## 2    Definitions

We consider the problem of supervised learning in an attribute based description language. The attributes (input variables) are assumed to be Boolean and every exemplar in the training set is labeled with a value that describes its class. Both algorithms try to maximize the mutual information between the network output and these labels.

Let variable $X$ take the values $\{x_1, x_2, ...x_n\}$ with probabilities $p(x_1), p(x_2)...p(x_n)$. The entropy of $X$ is given by $H(X) = -\sum_j p(x_j) \log p(x_j)$ and is a measure of the uncertainty about the value of $X$. The uncertainty about the value of $X$ when the value of another variable $Y$ is known is given by $H(X|Y) = -\sum_i p(y_i) \sum_j p(x_j|y_i) \log p(x_j|y_i)$.

The amount by which the uncertainty of $X$ is reduced when the value of variable $Y$ is known, $I(Y, X) = H(X) - H(X|Y)$ is called the mutual information between $Y$ and $X$. In this context, $Y$ will be a variable defined by the output of one or more nodes in the network and $X$ will be the target value specified in the training set.

# 3    Algorithms

## 3.1    Muesli - An algorithm for the design of multi-level logic networks

This algorithm derives the Boolean network by performing gradient descent in the mutual information between a set of nodes and the target values specified by the labels in the training set.

In the pseudo code description of the algorithm given in figure 1, the function $\mathcal{I}(S)$ computes the mutual information between the nodes in $S$ (viewed as a multi-valued variable) and the target output.

```
muesli(nlist) {
    nlist ← sort_nlist_by_I(nlist,1);
    sup ← 2;
    while (not_done(nlist) ∧ sup < max_sup) {
        act ← 0;
        do {
            act + +;
            success ← improve_mi(act, nlist, sup);
        } while (success = FALSE ∧ act < max_act);
        if (success = TRUE) {
            sup ← 2;
            while (success = TRUE)
                success ← improve_mi(act, nlist, sup);
        }
        else sup + +;
    }
}

improve_mi(act, nlist, sup) {
    nlist ← sort_nlist_by_I(nlist, act);
    f ← best_function(nlist, act, sup);
    if (I(nlist[1:act-1] ∪ f) > I(nlist[1:act])) {
        nlist ← nlist ∪ f;
        return(TRUE);
    }
    else return(FALSE);
}
```

Figure 1: Pseudo-code for the *Muesli* algorithm.

The algorithm works by keeping a list of candidate nodes, *nlist*, that initially contains only the primary inputs. The *act* variable selects which node in *nlist* is active. Initially, *act* is set to 1 and the node that provides more information about the output is selected as the active node. Function *improve_mi()* tries to combine the active node with other nodes as to increase the mutual information.

Except for very simple functions, a point will be reached where no further improve-

ments can be made for the single most *informative* node. The value of *act* is then increased (up to a pre-specified maximum) and *improve_mi* is again called to select auxiliary features using other nodes in *nlist* as the active node. If this fails, the value of *sup* (size of the support of each selected function) is increased until no further improvements are possible or the target is reached.

The function *sort_nlist_by_$\mathcal{I}$(nlist, act)* sorts the first *act* nodes in the list by decreasing value of the information they provide about the labels. More explicitly, the first node in the sorted list is the one that provides maximal information about the labels. The second node is the one that will provide more additional information after the first has been selected and so on.

Function *improve_mi()* calls *best_function(nlist, act, sup)* to select the Boolean function $f$ that takes as inputs node *nlist[act]* plus *sup*–1 other nodes and maximizes $\mathcal{I}(nlist[1:act-1] \cup f)$. When *sup* is larger than 2 it is unfeasible to search all $2^{2^{sup}}$ possible functions to select the desired one. However, given *sup* input variables, finding such a function is equivalent to selecting a partition[2] of the $2^{sup}$ points in the input space that maximizes a specific cost function. This partition is found using the Kernighan-Lin algorithm [Kernighan and Lin, 1970] for graph-partitioning.

Figure 2 exemplifies how the algorithm works when learning the simple Boolean function $f = ab + cde$ from a complete training set. In this example, the value of *sup* is always at 2. Therefore, only 2 input Boolean functions are generated.

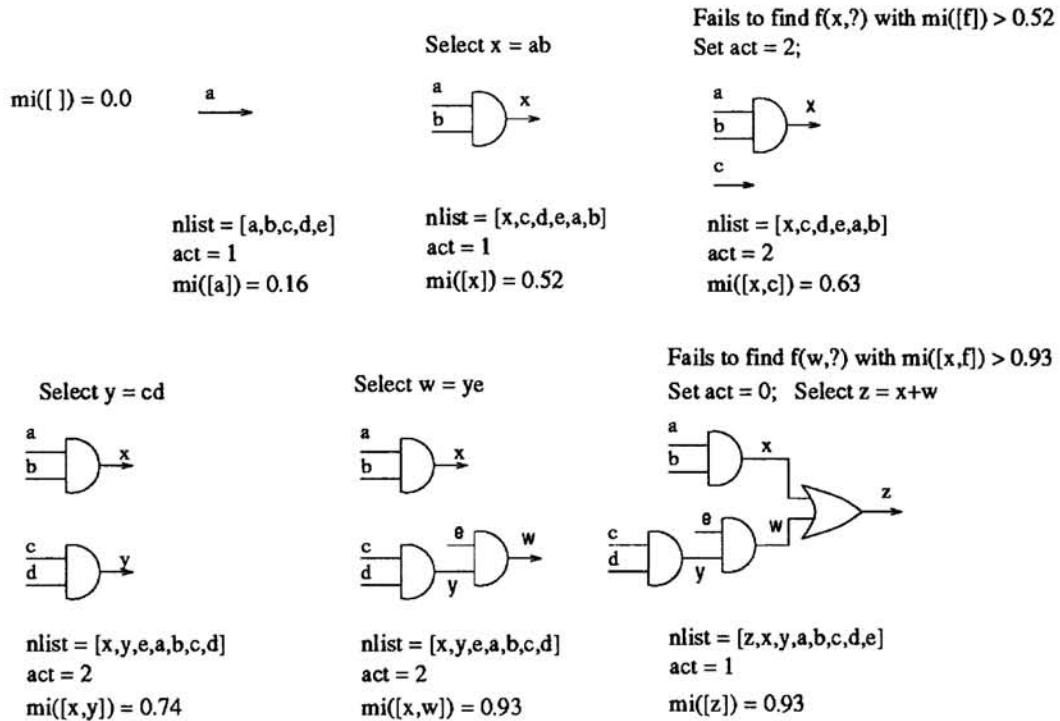

Figure 2: The *muesli* algorithm, illustrated

## 3.2   Fulfringe - a network generation algorithm based on decision trees

This algorithm uses binary decision trees [Quinlan, 1986] as the basic underlying representation. A binary decision tree is a rooted, directed, acyclic graph, where each terminal node (a node with no outgoing edges) is labeled with one of the possible output labels and each non-terminal node has exactly two outgoing edges labeled 0 and 1. Each non-terminal node is also labeled with the name of the attribute that is tested at that node. A decision tree can be used to classify a particular example by starting at the root node and taking, until a terminal is reached, the edge labeled with the value of the attribute tested at the current node.

Decision trees are usually built in a greedy way. At each step, the algorithm greedily selects the attribute to be tested as the one that provides maximal information about the label of the examples that reached that node in the decision tree. It then recurs after splitting these examples according to the value of the tested attribute.

*Fulfringe* works by identifying patterns near the fringes of the decision tree and using them to build new features. The idea was first proposed in [Pagallo and Haussler, 1990].

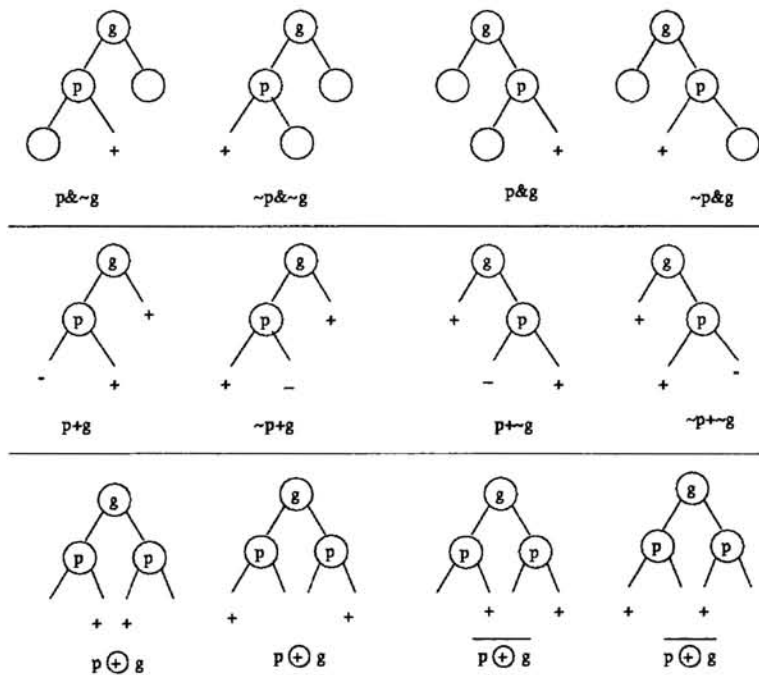

Figure 3: Fringe patterns identified by *fulfringe*

Figure 3 shows the patterns that *fulfringe* identifies. *Dcfringe*, proposed in [Yang *et al.*, 1991], identifies the patterns shown in the first two rows. These patterns correspond to 8 Boolean functions of 2 variables. Since there are only 10 distinct Boolean functions that depend on two variables[3], it is natural to add the patterns in the third row and identify all possible functions of 2 variables. As in *dcfringe* and *fringe*, these new composite features are added (if they have not yet been generated) to the list of available features and a new decision tree is built. The

process is iterated until a decision tree with only one decision node is built. The attribute tested at this node is a complex feature and can be viewed as the output of a Boolean network that matches the training set data.

### 3.3    Encoding multivalued outputs

Both *muesli* and *fulfringe* generate Boolean networks with a single binary valued output. When the target label can have more than 2 values, some encoding must be used. The prefered solution is to encode the outputs using an error correcting code [Dietterich and Bakiri, 1991]. This approach preserves most of the compactness of a digital encoding while beeing much less sensitive to errors in one of the output variables. Additionally, the Hamming distance between an observed output and the closest valid codeword gives a measure of the certainty of the classification. This can be used to our advantage in problems where a failure to classify is less serious than the output of a wrong classification.

## 4    Performance evaluation

To evaluate the algorithms, we selected a set of 11 functions of variable complexity. A complete description of these functions can be found in [Oliveira, 1994]. The first 6 functions were proposed as test cases in [Pagallo and Haussler, 1990] and accept compact *disjoint normal form* descriptions. The remaining ones accept compact multi-level representations but have large two level descriptions. The algorithms described in sections 3.1 and 3.2 were compared with the *cascade-correlation* algorithm [Fahlman and Lebiere, 1990] and a standard decision tree algorithm analog to ID3 [Quinlan, 1986]. As in [Pagallo and Haussler, 1990], the number of examples in the training set was selected to be equal to $\frac{1}{\epsilon}$ times the description length of the function under a fixed encoding scheme, where $\epsilon$ was set equal to 0.1. For each function, 5 training sets were randomly selected. The average accuracy for the 5 runs in an independent set of 4000 examples is listed in table 1.

Table 1: Accuracy of the four algorithms.

| Function | # inputs | # examples | Accuracy | | | |
|---|---|---|---|---|---|---|
| | | | muesli | fulfringe | ID3 | CasCor |
| dnf1 | 80 | 3292 | 99.91 | 99.98 | 82.09 | 75.38 |
| dnf2 | 40 | 2185 | 99.28 | 98.89 | 88.84 | 73.11 |
| dnf3 | 32 | 1650 | 99.94 | 100.00 | 89.98 | 79.19 |
| dnf4 | 64 | 2640 | 100.00 | 100.00 | 72.61 | 58.41 |
| xor4_16 | 16 | 1200 | 98.35 | 100.00 | 75.20 | 99.91 |
| xor5_32 | 32 | 4000 | 60.16 | 100.00 | 51.41 | 99.97 |
| sm12 | 12 | 1540 | 99.90 | 100.00 | 99.81 | 98.98 |
| sm18 | 18 | 2720 | 100.00 | 99.92 | 91.48 | 91.30 |
| str18 | 18 | 2720 | 100.00 | 100.00 | 94.55 | 92.57 |
| str27 | 27 | 4160 | 98.64 | 99.35 | 94.24 | 93.90 |
| carry8 | 16 | 2017 | 99.50 | 98.71 | 96.70 | 99.22 |
| Average | | | 95.97 | 99.71 | 85.35 | 87.45 |

The results show that the performance of *muesli* and *fulfringe* is consistently su-

perior to the other two algorithms. *Muesli* performs poorly in examples that have many *xor* functions, due the greedy nature of the algorithm. In particular, *muesli* failed to find a solution in the alloted time for 4 of the 5 runs of *xor5_32* and found the exact solution in only one of the runs.

ID3 was the fastest of the algorithms and Cascade-Correlation the slowest. *Fulfringe* and *muesli* exhibited similar running times for these tasks. We observed, however, that for larger problems the runtime for *fulfringe* becomes prohibitively high and *muesli* is comparatively much faster.

# 5    Applications

To evaluate the techniques described in real problems, experiments were performed in two domains: noisy image reconstruction and handwritten character recognition. The main objective was to investigate whether the approach is applicable to problems that are not known to accept a compact Boolean network representation. The outputs were encoded using a 15 bit Hadamard error correcting code.

## 5.1    Image reconstruction

The speed required by applications in image processing makes it a very interesting field for this type of approach. In this experiment, 16 level gray scale images were corrupted by random noise by switching each bit with 5% probability. Samples of this image were used to train a network in the reconstruction of the original image. The training set consisted of 5x5 pixel regions of corrupted images (100 binary variables per sample) labeled with the value of the center pixel. Figure 4 shows a detail of the reconstruction performed in an independent test image by the network obtained using *fulfringe*.

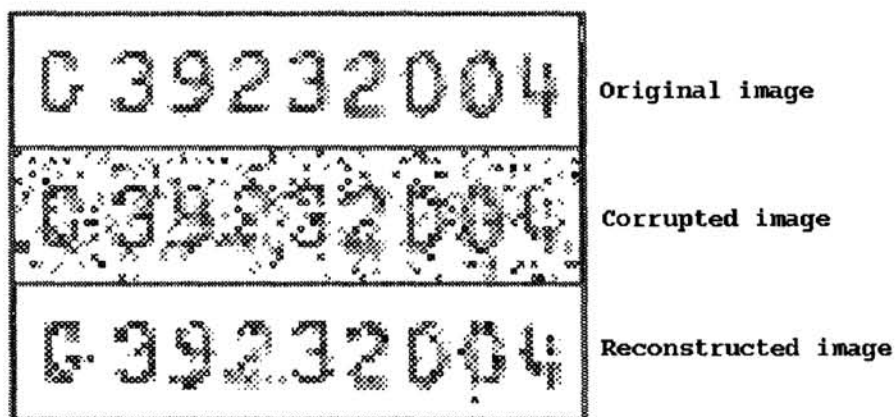

Figure 4: Image reconstruction experiment

## 5.2    Handwritten character recognition

The NIST database of handwritten characters was used for this task. Individually segmented digits were normalized to a 16 by 16 binary grid. A set of 53629 digits was used for training and the resulting network was tested in a different set of 52467

digits. Training was performed using *muesli*. The algorithm was stopped after a pre-specified time (48 hours on a DECstation 5000/260) ellapsed. The resulting network was placed and routed using the TimberWolf [Sechen and Sangiovanni-Vincentelli, 1986] package and occupies an area of 78.8 sq. mm. using $0.8\mu$ technology.

The accuracy on the test set was 93.9%. This value compares well with the performance obtained by alternative approaches that use a similarly sized training set and little domain knowledge, but falls short of the best results published so far. Ongoing research on this problem is concentrated on the use of domain knowledge to restrict the search for compact networks and speed up the training.

## Acknowledgements

This work was supported by Joint Services Electronics Program grant F49620-93-C-0014.

## Footnotes

[1]Up to some specified level.

[2] A single output Boolean function is equivalent to a partition of the input space in two sets.

[3]The remaining 6 functions of 2 variables depend on only one or none of the variables.

## References

[Blumer *et al.*, 1987] A. Blumer, A. Ehrenfeucht, D. Haussler, and M. Warmuth. Occam's razor. *Information Processing Letters*, 24:377–380, 1987.

[Dietterich and Bakiri, 1991] T. G. Dietterich and G. Bakiri. Error-correcting output codes: A general method for improving multiclass inductive learning programs. In *Proceedings of the Ninth National Conference on Artificial Intelligence (AAAI-91)*, pages 572–577. AAAI Press, 1991.

[Fahlman and Lebiere, 1990] S.E. Fahlman and C. Lebiere. The cascade-correlation learning architecture. In D.S. Touretzky, editor, *Advances in Neural Information Processing Systems*, volume 2, pages 524–532, San Mateo, 1990. Morgan Kaufmann.

[Garey and Johnson, 1979] M.R. Garey and D.S. Johnson. *Computers and Intractability: A Guide to the Theory of NP-Completeness*. Freeman, New York, 1979.

[Kernighan and Lin, 1970] B. W. Kernighan and S. Lin. An efficient heuristic procedure for partitioning graphs. *The Bell System Technical Journal*, pages 291–307, February 1970.

[Murray and Smith, 1988] Alan F. Murray and Anthony V. W. Smith. Asynchronous vlsi neural networks using pulse-stream arithmetic. *IEEE Journal of Solid-State Circuits*, 23:3:688–697, 1988.

[Oliveira, 1994] Arlindo L. Oliveira. *Inductive Learning by Selection of Minimal Representations*. PhD thesis, UC Berkeley, 1994. In preparation.

[Pagallo and Haussler, 1990] G. Pagallo and D. Haussler. Boolean feature discovery in empirical learning. *Machine Learning*, 1, 1990.

[Quinlan, 1986] J. R. Quinlan. Induction of decision trees. *Machine Learning*, 1:81–106, 1986.

[Rissanen, 1986] J. Rissanen. Stochastic complexity and modeling. *Annals of Statistics*, 14:1080–1100, 1986.

[Sechen and Sangiovanni-Vincentelli, 1986]
Carl Sechen and Alberto Sangiovanni-Vincentelli. TimberWolf3.2: A new standard cell placement and global routing package. In *Proceedings of the 23rd Design Automation Conference*, pages 432–439, 1986.

[Yang *et al.*, 1991] D. S. Yang, L. Rendell, and G. Blix. Fringe-like feature construction: A comparative study and a unifying scheme. In *Proceedings of the Eight International Conference in Machine Learning*, pages 223–227, San Mateo, 1991. Morgan Kaufmann.